# A Global Structural EM Algorithm for a Model of Cancer Progression

**Ali Tofigh**
School of Computer Science
McGill Centre for Bioinformatics
McGill University, Canada
ali.tofigh@mcgill.ca

**Erik Sjölund**
Stockholm Bioinformatics Center
Stockholm University, Sweden
erik.sjölund@sbc.su.se

**Mattias Höglund**
Department of Oncology
Lund University, Sweden
mattias.hoglund@med.lu.se

**Jens Lagergren**
Science for Life Lab
Swedish e-Science Research Center
Stockholm Bioinformatics Center
School of Computer Science and Communication
KTH Royal Institute of Technology, Sweden
jensl@csc.kth.se

## Abstract

Cancer has complex patterns of progression that include converging as well as diverging progressional pathways. Vogelstein's path model of colon cancer was a pioneering contribution to cancer research. Since then, several attempts have been made at obtaining mathematical models of cancer progression, devising learning algorithms, and applying these to cross-sectional data. Beerenwinkel *et al.* provided, what they coined, EM-like algorithms for Oncogenetic Trees (OTs) and mixtures of such. Given the small size of current and future data sets, it is important to minimize the number of parameters of a model. For this reason, we too focus on tree-based models and introduce Hidden-variable Oncogenetic Trees (HOTs). In contrast to OTs, HOTs allow for errors in the data and thereby provide more realistic modeling. We also design global structural EM algorithms for learning HOTs and mixtures of HOTs (HOT-mixtures). The algorithms are global in the sense that, during the M-step, they find a structure that yields a global maximum of the expected complete log-likelihood rather than merely one that improves it. The algorithm for single HOTs performs very well on reasonable-sized data sets, while that for HOT-mixtures requires data sets of sizes obtainable only with tomorrow's more cost-efficient technologies.

## 1  Introduction

In the learning literature, there are several previous results on learning probabilistic tree models, including various Expectation Maximization-based inference algorithms. In [1], trees were considered where the vertices were associated with observable variables and an efficient algorithm for finding a globally optimal Maximum Likelihood (ML) solution was described. Subsequently, [2] presented a structural Expectation Maximization (EM) algorithm for finding the ML mixture of trees as well as MAP solutions with respect to several priors.

There are three axes along which it is natural to compare these as well as other results. The first axis is the type of dependency structure allowed. The second axis is the type of variables used—

observable only or hidden and observable—and the type of relations they can have. The third axis is the type of inference algorithms that are known for the model.

It is interesting in relation to the present result to ask in what respect the structural EM algorithm of [3] constitutes an improvement when compared with Friedman's earlier structural EM algorithm [4]. In fact, it may seem like the former constitutes no improvement at all, since the latter is concerned with more general dependency structures. Notice, however, that it is customary to distinguish between EM algorithms and generalized EM algorithms for inferring numerical parameters, the difference being that in the M-step of the former, parameters are found that maximize the expected complete log-likelihood, whereas in the latter, parameters are found that merely improve it. As Friedman points out in his article on the Bayesian Structural EM algorithm [4], the same distinction can be made regarding the maximization over structures. Clearly, it would be convenient to use the same terminology for structural EM algorithms as for ordinary EM algorithms. However, the distinction is often not made for structural EM algorithms and even researchers that consider themselves experts in the field seem to be unaware of it. For this reason, we define *global* structural EM algorithms to be EM algorithms that in the M-step find a structure yielding a global maximum of the expected complete log-likelihood (as opposed to a structure that merely improves it). Equipped with this definition, we note that the phylogeny algorithm of [3] is a global structural EM algorithm in contrast to the earlier algorithm [4]. Another example of a global structural EM algorithm is the learning algorithm for trees with hidden variables presented in [5].

In an effort to provide mathematical models of cancer progression, Desper *et al.* introduced the Oncogenetic Tree model where observable variables corresponding to aberrations are associated with vertices of a tree [6]. They then proceeded to show that an algorithm based on Edmonds's optimum branching algorithm will, with high probability, correctly reconstruct an Oncogenetic Tree $\mathcal{T}$ from sufficiently long series of data generated from $\mathcal{T}$.

The Oncogenetic Tree model suffers from two problems; monotonicity—an aberration associated with a child cannot occur unless the aberration associated with its parent has occurred—and limited-structure—compared to a network, the tree structure severely limits the sets of progressional paths that can be modeled. In an attempt to remedy these problems, the Network Aberration Model was proposed [7, 8]. However, the computational problems associated with these network models are hard; for instance, no efficient EM algorithm for training is yet known. In another attempt, Beerenwinkel *et al.* used mixtures of Oncogenetic Trees to overcome the problem of limited-structure, but without removing the monotonicity and only obtaining an algorithm with an EM-like structure that has not been proved to deliver a locally optimal maximum-likelihood solution [9, 10, 11].

Beerenwinkel and coworkers used Conjunctive Bayesian Networks (CBNs) to model cancer progression [12, 13]. In order to overcome the limited ability of CBNs to model noisy biological data, [14] introduced the hidden CBN model. A hidden CBN can be obtained from a CBN by considering each variable in the CBN to be hidden and associating an observable variable with each hidden variable. The hidden CBN also has a common error parameter specifying the probability that any individual observable variable differs from its associated hidden variable. In a hidden CBN, values are first generated for the hidden variables, and then, the observable variables obtain values based both on the hidden variables and the error parameter.

We present the Hidden-variable Oncogenetic Tree (HOT) model where a hidden and an observable variable are associated with each vertex of a rooted directed tree. The value of the hidden variable indicates whether or not the tumor progression has reached the vertex (a value of one means that cancer progression has reached the vertex and zero that it has not), while the value of the observable variable indicates whether a specific aberration has been detected (a value of one represents detection and zero the opposite). This interpretation provides several relations between the variables in a HOT. An asymmetric relation is required between the hidden variables associated with the two endpoints of an arc of the directed tree. Because of this asymmetry, the global structural EM algorithm that we derive for the HOT ML problem cannot, in contrast to many of the above mentioned algorithms, be based on a maximum spanning tree algorithm and is instead based on the optimal branching algorithm [15, 16, 17]. Having so rectified the monotonicity problem, we proceed to obtain a model allowing for a higher degree of structural variation by introducing mixtures of HOTs (HOT-mixtures) and, in contrast to Beerenwinkel et al., we derive a proper structural EM algorithm for training these.

In the near future, multiple types of high throughput (HTP) data will be available for large collections of tumors, providing great opportunities as well as computational challenges for progression model inference. One of the main motivations for our models and inference methods is that they enable analysis of future HTP-data, which most likely will require the ability to handle large numbers of mutational events. In this paper, however, we apply our methods to cytogenetic data for colon and kidney cancer, mostly due to the availability of cytogenetic data for large numbers of tumors provided by the Mitelman database [18].

## 2 HOTs and the novel global structural EM algorithm

### 2.1 Hidden-variable Oncogenetic Trees

We will denote the set of observed data points $D$ and an individual data point $X$. In Section 3, we will apply our methods to CNA, i.e., a data point will be a set of observed copy number abberations, but in general, more complex events can be used.

A *rooted directed tree* $T$ consists of a set of vertices, denoted $V(T)$ and a set of arcs denoted $A(T)$. An arc $\langle u, v \rangle$ is directed from the vertex $u$ called its *tail* towards the vertex $v$ called its *head*. If there is an arc with tail $p$ and head $u$ in a directed tree $T$, then $p$ is called the parent of $u$ in $T$ and denoted $p(u)$ (the tree $T$ will be clear from context).

An OT is a rooted directed tree where an aberration associated with each vertex and a probability associated with each arc. One can view an OT as generating a set of aberrations by first visiting the root and then continuing towards the leaves (preorder) visiting each vertex with the probability of its incoming arc if the parent has been visited, and with probability zero if the parent has not been visited. The result of the progression is the set of aberrations associated with the visited vertices.

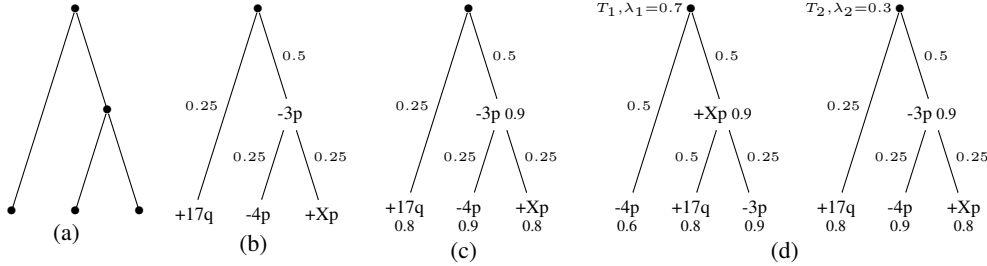

Figure 1: **(a)** *A rooted directed tree with the root at the top. All arcs are directed downwards, i.e., away from the root.* **(b)** *An OT with probabilities associated with arcs and CNAs associated with vertices.* **(c)** *A HOT with probabilities associated with arcs (indicating the probability that the hidden variable associated with the head of the arc receives the value 1 conditioned that the hidden variable associated with the tail has this value), and CNAs as well as probabilities associated with vertices (indicating the probability that the observable variable associated with the vertex receives the value 1 conditioned that the hidden variable associated with the vertex has received this value).* **(d)** *A HOT-mixture consisting of two HOTs. The mixing probability for $T_1$ is 0.7 and that for $T_2$ is 0.3. So with probability 0.7 a synthetic tumor is generated from $T_1$ and otherwise one is generated from $T_2$.*

In Figure 1(b), an OT for CNA is depicted (aberrations are written in the standard notation for CNAs in cytogenetic data, i.e., each represents a duplication (+) or deletion (-) of a specific chromosomal region). Notice that an aberration associated with a vertex cannot occur unless the aberration associated with its parent has occurred. For instance, the set $\{+Xp, +17q\}$ cannot be generated by the OT in Figure 1(b). In a data-modeling context, this is highly undesirable as data is typically noisy and is bound to contain both false positives and negatives. Our HOT model does not suffer from this problem.

A Hidden-variable Oncogenetic Tree (HOT) is a directed tree where, just like OTs, each vertex represents a specific aberration. Unlike OTs however, the progression of cancer is modeled with hidden variables associated with vertices and conditional probabilities associated with the arcs. The observation of the aberrations (the data) are modeled with a different set of random variables whose values are conditioned on the hidden variables.

Formally, a Hidden-variable Oncogenetic Tree (HOT) is a pair $\mathcal{T} = (T, \Theta)$ where:

1. $T$ is a rooted directed tree and $\Theta$ consists of two conditional probability distributions, $\theta_X(u)$ and $\theta_Z(u)$, for each vertex $u$;

2. two random variables are associated with each vertex: an observable variable $X(u)$ and a hidden variable $Z(u)$, each assuming the values 0 or 1;

3. the hidden variable associated with the root, $Z(r)$, is defined to have a value of one;

4. for each non-root vertex $u$, $\theta_Z(u)$ is a conditional probability distribution on $Z(u)$ conditioned by $Z(p(u))$ satisfying $\Pr[Z(u) = 1 | Z(p(u)) = 0] = \epsilon_Z(u)$; and

5. for each non-root vertex $u$, $\theta_X(u)$ is a conditional probability distribution on $X(u)$ conditioned by $Z(u)$ satisfying $\Pr[X(u) = 1 | Z(u) = 0] = \epsilon_X(u)$.

With respect to (4), one might argue that $\Pr[Z(u) = 1 | Z(p(u)) = 0]$ should be zero, since if the progression has not reached $p(u)$ it should not be able to proceed to $u$. However, the derivation and implementation of the EM algorithm depends on the non-zero value of this probability for much the same reasons that people use pseudo-counts [19], namely, once a parameter receives the value 0 in an EM algorithm for training, it will subsequently not be changed. Moreover, $\epsilon_Z$ has a natural interpretation: it corresponds to a small probability of spontaneous mutations occurring independently from the overall progressional path that the disease is following. Similar arguments apply to (5) where we interpret $\epsilon_X$ as the small probability of falsely detecting an aberration that is not actually present (corresponding to a false positive test).

We note here that it is possible to have CPDs where $X(u)$ and $Z(u)$ depend on both $X(p(u))$ and $Z(p(u))$, and even to let $X(u)$ depend on all three of $Z(u)$, $X(p(u))$, and $Z(p(u))$. We note here that our arguments can easily be extended to cover these cases, although we will not consider them further in the following text. Figure 1(c) shows an example of a HOT where $\epsilon_Z$ and $\epsilon_X$ have been omitted for clarity.

## 2.2 The novel global structural EM algorithm for HOTs

We have derived a global structural Expectation Maximization (EM) algorithm for inferring HOTs from data. According to standard EM theory [20], such an algorithm is obtained if there is a procedure that given a HOT $\mathcal{T}$ finds a HOT $\mathcal{T}'$ that maximizes the so-called complete log-likelihood (also known as the $Q$-term):

$$Q(\mathcal{T}'; \mathcal{T}) = \sum_{X \in D} \sum_Z \Pr[Z|X, \mathcal{T}] \log \Pr[Z, X|\mathcal{T}'].$$

The likelihood of $\mathcal{T}'$ is guaranteed to be at least as high as $\mathcal{T}$, which immediately leads to an iterative procedure. In standard EM, the $Q$-term is maximized only over the parameters of a model, in our case the conditional probabilities, leaving the structure, i.e., the directed tree, unchanged. Friedman *et al.* [3] extended the use of EM algorithms from the standard parameter estimation to also finding an optimal structure. In their case, the probabilistic model was reversible and the tree that maximized the expected complete log-likelihood could be obtained using a maximum spanning tree algorithm. In our case, the pair-wise relations between hidden variables are asymmetric and a maximum spanning tree algorithm cannot be used. However, as we show below, the $Q$-term can be maximized by instead using Edmonds's optimal branching algorithm.

When dealing with mixtures of HOTs in later sections, we will need to maximize the *weighted* version of the $Q$-term, which we introduce already here:

$$Q_f(\mathcal{T}'; \mathcal{T}) = \sum_{X \in D} \sum_Z f(X)\Pr[Z|X, \mathcal{T}] \log \Pr[Z, X|\mathcal{T}'], \tag{1}$$

where $f$ is a weight function on the data points in $D$ and can be computed in constant time.

By expanding and rearranging the terms in (1) (see the appendix), it can be shown that $Q_f(\mathcal{T}'; \mathcal{T})$ equals

$$\sum_{\langle u,v \rangle \in A(T')} \sum_{a,b \in \{0,1\}} \sum_{X \in D} f(X)\Pr[Z(v) = a, Z(u) = b|X, \mathcal{T}] \log \Pr[Z(v) = a|Z(u) = b, \theta'_Z(u)]$$

$$+ \sum_{\langle u,v\rangle \in A(T)} \sum_{\sigma,a\in\{0,1\}} \sum_{X\in D:X(u)=\sigma} f(X)\mathrm{Pr}[Z(v)=a|X,\mathcal{T}]\log\mathrm{Pr}[X(v)=\sigma|Z(v)=a,\theta'_X(u)].$$

As long as the directed tree $T'$ is fixed, the standard EM methodology (see for instance [19]) can be used to find the $\Theta'$ that maximizes $Q_f(T',\Theta';\mathcal{T})$ as follows. First, let

$$A_u(a,b) = \sum_{X\in D} f(X)\mathrm{Pr}[Z(u)=a, Z(p'(u))=b|X,\mathcal{T}] \tag{2}$$

and

$$B_u(\sigma,a) = \sum_{X\in D:X(u)=\sigma} f(X)\mathrm{Pr}[Z(u)=a|X,\mathcal{T}]. \tag{3}$$

Then the $\Theta'$ that, for a fixed $T'$, maximizes $Q_f(\mathcal{T}';\mathcal{T})$ (i.e. $Q_f(T',\Theta';\mathcal{T})$) is given by

$$\mathrm{Pr}[Z(u)=a|Z(p'(u))=b,\theta'_Z(u)] = A_u(a,b)/(\sum_{a\in\{0,1\}} A_u(a,b))$$

and

$$\mathrm{Pr}[X(u)=\sigma|Z(u)=a,\theta'_Z(u)] = B_u(\sigma,a)/(\sum_{\sigma\in\{0,1\}} B_u(\sigma,a)).$$

The time required for computing the right hand sides of (2) and (3) is $O(n^2)$, where $n$ is the number of aberrations (The probabilities $\mathrm{Pr}[Z(u)=a, Z(v)=b|X,\mathcal{T}]$ can be computed using techniques analogous to those appearing in [3]).

For each arc $\langle p,u\rangle$ of $T'$, using the CPDs defined above, we define the weight of the arc, specific to this tree to be

$$\sum_{a,b\in\{0,1\}} \sum_{X\in D} f(X)\mathrm{Pr}[Z(u)=a, Z(p'(u))=b|X,\mathcal{T}]\log\mathrm{Pr}[Z(u)=a|Z(p'(u))=b,\theta'_Z(u)]$$

$$+ \sum_{b\in\{0,1\}} \sum_{X\in D} f(X)\mathrm{Pr}[Z(u)=a|X,\mathcal{T}]\log\mathrm{Pr}[X(u)|Z(u)=a,\theta'_X(u)].$$

We now make two important observations from which it follows how to maximize the weighted expected complete log-likelihood over all directed trees. First, notice that if two directed trees $T'$ and $T''$ have a common arc $\langle p,u\rangle$, then this arc has the same weight in these two trees (since the weights on the arc does not depend on any other arc in the tree). Let $G$ be the directed, complete, and arc-weighted graph with the same vertex set as the tree $T$, and with arc weights given by the above expression.

An optimal arborescence of a directed graph is a rooted directed tree on the same set of vertices as the directed graph, i.e., a subgraph that has exactly one directed path from one specified vertex called the root to any other vertex, and has maximum arc weight sum among all such rooted directed trees. For any arborescence $T'$ of $G$, the sum of the arc weights equals, by the construction of $G$, the maximum value of $Q_f(T',\Theta';\mathcal{T})$ over all $\Theta'$. From this follows that, a (spanning) directed tree $T'$ is an optimal arborescence of $G$ if and only if $T'$ maximizes the $Q_f$ term. And so, applying Edmonds's algorithm to $G$ gives the desired directed tree. Tarjan's implementation of Edmonds's algorithm runs in quadratic time [15, 16, 17]. Hence, the total running time for the algorithm is $O(|D|\cdot n^2)$.

## 2.3 HOT-mixtures

In this section we extend our model to HOT-mixtures by including an initial random choice of one of several HOTs and letting the final outcome be generated by the chosen HOT. We will also obtain an EM-based model-training algorithm for HOT-mixtures by showing how to optimize the expected complete log-likelihoods for HOT-mixtures. Formally, we will use $k$ HOTs $\mathcal{T}_1,\ldots,\mathcal{T}_k$ and a random mixing variable $I$ that takes on values in $1,\ldots,k$. The probability that $I=i$ is denoted $\lambda_i$ and $\lambda = (\lambda_1,\ldots,\lambda_k)$ is a vector of parameters of the model in addition to those of the HOTs ($\lambda_1,\ldots,\lambda_k$ are constrained to sum to 1). The following notation is convenient

$$\gamma_i(X) = \mathrm{Pr}[I=i|X,M] = \frac{\lambda_i\mathrm{Pr}[X|\mathcal{T}_i]}{\sum_{j\in[k]} \lambda_j\mathrm{Pr}[X|\mathcal{T}_j]}.$$

For a HOT-mixture, the expected complete log-likelihood can be expressed as follows

$$\sum_{X \in D} \sum_{Z,I} \Pr[Z, I | X, M] \log \Pr[Z, I, X | M'].$$ (4)

Using standard EM methodology, it is possible to show that (4) can be maximized by independently maximizing

$$\sum_{i \in [k]} \sum_{X \in D} \gamma_i(X) \log(\lambda_i')$$ (5)

and, for each $i = 1, \ldots, k$, maximizing

$$\sum_{X \in D} \sum_{Z} \Pr[Z | X, \mathcal{T}_i] \gamma_i(X) \log(\Pr[Z, X | \mathcal{T}_i'])$$ (6)

Finding a $\lambda' = \lambda_1', \ldots, \lambda_k'$ maximizing (5) is straightforward (see for instance [19]) and, for each $i = 1, \ldots, k$, finding a $\mathcal{T}_i'$ that maxmizes the weighted $Q$-term in (6) can be done as described in the previous subsections (with $\gamma_i(X)$ weighting the data points).

## 3  Results

In this section, we report results obtained by applying our algorithms to synthetic and cytogenetic cancer data.

In the standard version of the EM algorithm, there are four parameters per edge of a HOT. The number of parameters can be reduced by letting some parameters be global, e.g., by letting $\epsilon_x(u) = \epsilon_x(u')$ for all vertices $u$ and $u'$. There are three parameters whose global estimation is desirable: $\epsilon_x$, $\epsilon_Z$, and $\Pr[X(u) = 0 | Z(u) = 1]$. However, for technical reasons, requiring that $\epsilon_z$ be global makes it impossible to derive an EM algorithm. Therefore, we will distinguish between two different versions of the algorithm: one with *free parameters* and one with *global parameters*. The free parameter version then corresponds to the standard EM algorithm, while the global parameter version corresponds to letting $\epsilon_x$ and $\Pr[X(u) = 0 | Z(u) = 1]$ be global. When evaluating the global parameter version of the algorithm using synthetic data, we will follow the convention of letting all three error parameters be global when generating data.

Other conventions used for all the tests described here include the following. We enforce an upper limit of 0.5 on $\epsilon_z$ and $\epsilon_x$. Also, for each data set, we first run the algorithm on a set of randomly generated start HOTs or start HOT-mixtures for 10 iterations. The HOT or HOT-mixture that results in the best likelihood is then run until convergence. Unless stated otherwise, the number of start trees and mixtures is 100.

### 3.1  Tests on Synthetic Data sets

#### 3.1.1  Single HOTs

We generated random HOTs with 10, 25, and 40 vertices with parameters on the edges chosen uniformly in the intervals

$$\Pr[Z(u) = 1 | Z(p(u)) = 1] \in [0.1, 1.0],$$ (7)
$$\Pr[X(u) = 0 | Z(u) = 1], \epsilon_x, \epsilon_z \in [0.01, q],$$ (8)

where $q \in \{0.05, 0.10, 0.25, 0.50\}$. For each combination, we generated 100 HOTs for a total of $3 \times 4 \times 100 = 1200$ HOTs. Figure 2 shows the result of our experiments on synthetic data. An edge of the generated HOT connecting one specific aberration to another is considered to have been correctly recovered if the HOT obtained from the algorithm connects the same two aberrations in the same direction. We also compared the performance of our algorithms with that of *Mtreemix* by Beerenwinkel *et al* [11]. The generated data from our single HOTs were passed to Mtreemix and the same criteria as above were used to detect correctly recovered edges (no special options were set when running Mtreemix on data generated with global parameters since no distinction between global and free parameters can be made on oncogenetic trees). Mtreemix outperforms our methods when the HOTs and the error parameters are small, while our algorithms outperform Mtreemix significantly as the size of the HOTs *or* error parameters become larger.

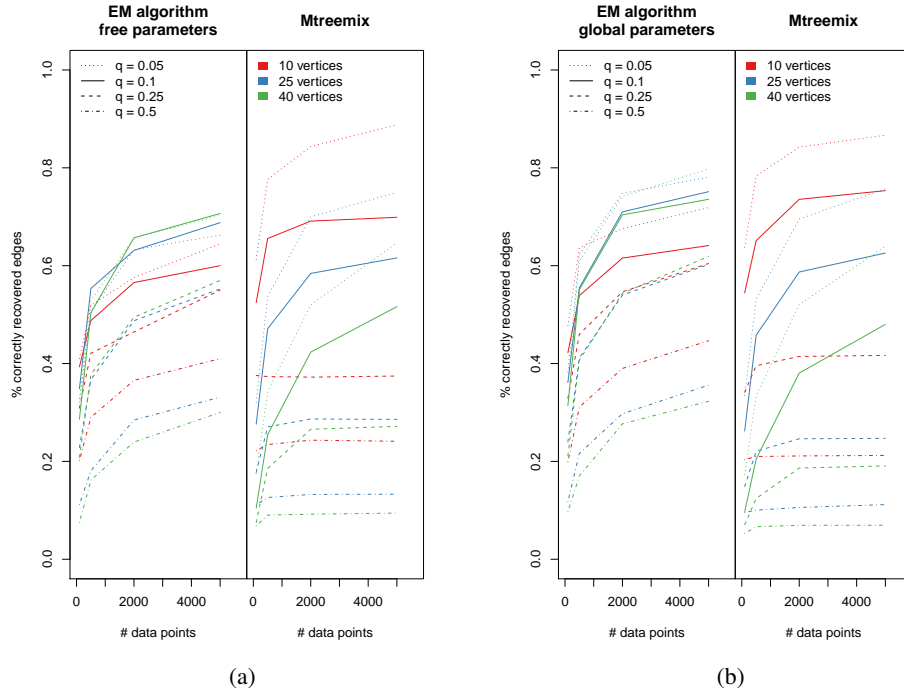

Figure 2: Histograms showing the mean percentage of edges that were correctly recovered by the algorithm for the free parameter case together with error bars showing one standard deviation.

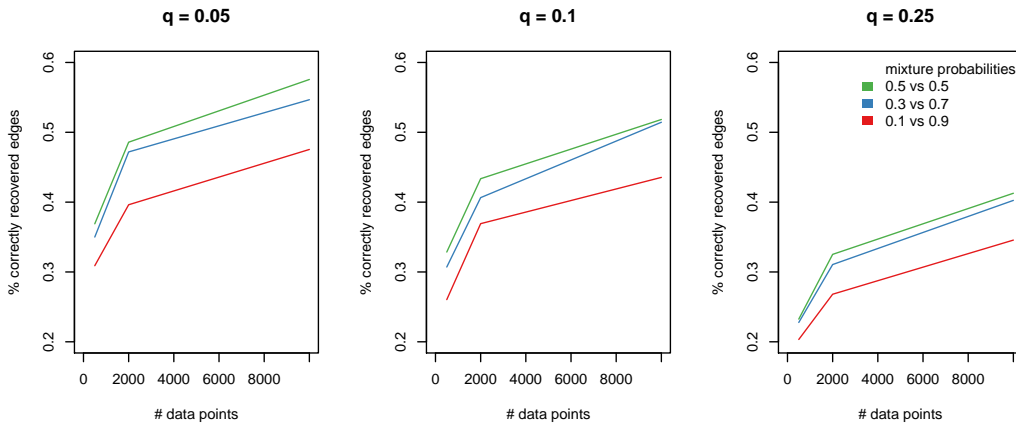

Figure 3: Histograms showing proportion of edges correctly recovered by the EM algorithm for HOT-mixtures with global parameters on two HOTs with 25 vertices each. Each bar represents 100 mixtures. Error bars show one standard deviation.

### 3.1.2 HOT Mixtures

We also tested the ability of the algoriithm to recover a mixture of two HOTs. The results are shown in 3. When measuring the number of correctly recovered edges, the following procedure was used. Each HOT produced from the algorithm was compared to each HOT from which the data was generated, and the number of correctly recovered edges was noted. The best way of matching the two HOTs produced from the algorithm with the two original HOTs was then determined. Two

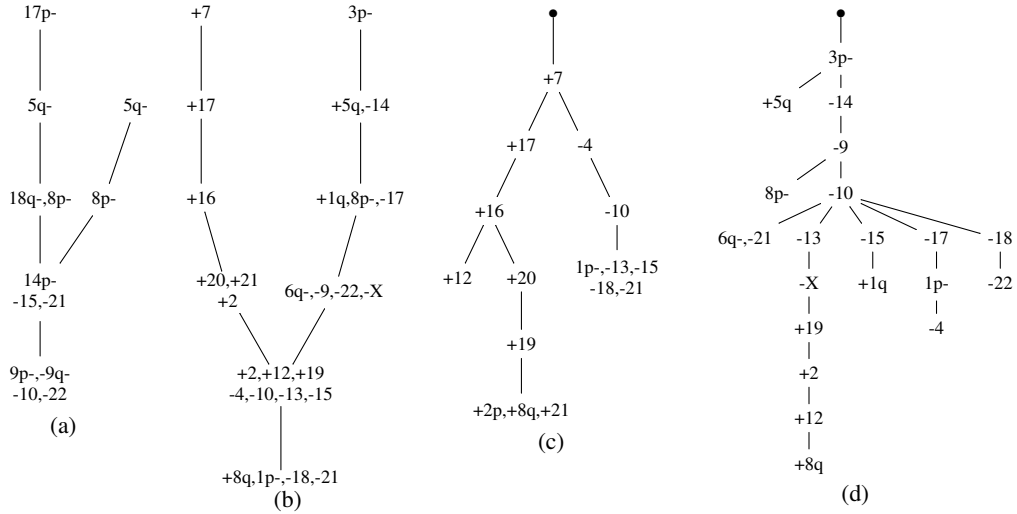

Figure 4: **HOTs obtained from RCC data.** *(a) shows an adapted version of the pathways for CC data published in [21]. (b) is a figure adapted from [22] showing the pathways obtained from statistical analysis of RCC data. (c) and (d) are the HOTs we obtained from the RCC data using only aberrations on the left and right pathways in (b), respectively. Notice the high level of agreement between the root-to-leaf paths in the recovered HOTs with those in (b).*

features can clearly be distinguished: the results improve as the size of the data increases, and the algorithm performs better when the HOTs have equal probability in the mixture.

## 3.2 Tests on Cancer Data

Our cytogenetic data for colon (CC) and kidney (RCC) cancer consist of 512 and 998 tumors, respectively. The data consist of measurements on 41 common aberrations (18 gains, 23 losses) for CC and 28 (13 gains, 15 losses) for RCC. The data have previously been analyzed in [21] and [22] resulting in suggested pathways of progression. These analyses were based on Principal Component Analysis (PCA) performed on correlations between aberrations and a statistical measure called *time of occurrence* (TO) which measures how early or late an aberration occurs during progression. The aberrations were then clustered based on the PCA and each cluster was manually formed into a pathway (based on PCA and TO). One advantage of our approach is that we are able to replace the manual curation by automated computational steps. Another advantage is that our models assign probabilities to data and the different models can therefore be compared objectively.

We expect $\epsilon_z$ and $\epsilon_x$ to be small in the type of data that we are using. We obtained the $n$ most correlated aberrations in our CC data, for $n \in \{4, \ldots, 11\}$, and tested different upper limits on $\epsilon_z$ and $\epsilon_x$. The best correspondence to previously published analyses of the data was found when $\epsilon_z \leq 0.25$ and $\epsilon_x \leq 0.01$ by counting the number of *bad edges*. A bad edge is one that contradicts the partial ordering given by the pathways described in [21], of which the relevant part is shown in the Figure 4(a).

Having found upper limits that work well on the CC data, we applied the algorithm with these upper bounds to the RCC data. The earlier analyses in [22] strongly suggests that two HOTs are required to model the RCC data. Given that our mixture model appears, from our tests on synthetic data, to require substantially more data points to recover the underlying HOTs in a satisfactory manner, we used the results of the analysis in [22] to divide the aberrations into two (overlapping) clusters for which we created HOTs separately. These HOTs can be seen in Figure 4(c) and 4(d) and they show very good agreement to the pathways from [22] shown in Figure 4(b). For instance, each root-to-leaf path in the HOT of Figure 4(c) agrees perfectly with the pathway shown in Figure 4(b).

## References

[1] C. Chow and C. Liu. Approximating discrete probability distributions with dependence trees. *IEEE Trans Inform Theor*, 14(3):462–467, 1968.

[2] M. Meila and M.I. Jordan. Learning with mixtures of trees. *J Mach Learn Res*, 1(1):1–48, 2000.

[3] N. Friedman, M. Ninio, I. Pe'er, and T. Pupko. A structural em algorithm for phylogenetic inference. *J Comput Biol*, 9(2):331–353, 2002.

[4] N. Friedman. The bayesian structural em algorithm. In *Proceedings of the Conference on Uncertainty in Artificial Intelligence*, pages 129–138. Morgan Kaufmann, 1998.

[5] P Leray and O François. Bayesian network structural learning and incomplete data. *Proceedings of the International and Interdisciplinary Conference on Adaptive Knowledge Representation and Reasoning (AKRR 2005)*, pages 33–40, 2005.

[6] R. Desper, F. Jiang, O.P. Kallioniemi, H. Moch, C.H. Papadimitriou, and A.A. Schaffer. Inferring tree models for oncogenesis from comparative genome hybridization data. *J Comput Biol*, 6(1):37–51, 1999.

[7] M. Hjelm, M. Höglund, and J. Lagergren. New probabilistic network models and algorithms for oncogenesis. *J Comput Biol*, 13(4):853–865, May 2006.

[8] M.D. Radmacher, R. Simon, R. Desper, R. Taetle, A.A. Schaffer, and M.A. Nelson. Graph models of oncogenesis with an application to melanoma. *J Theor Biol*, 212(4):535–48, Oct 2001.

[9] N. Beerenwinkel, J. Rahnenfuhrer, M. Daumer, D. Hoffmann, R. Kaiser, J. Selbig, and T. Lengauer. Learning multiple evolutionary pathways from cross-sectional data. *J Comput Biol*, 12(6):584–598, Jul 2005.

[10] J. Rahnenfuhrer, N. Beerenwinkel, W.A. Schulz, C. Hartmann, A. von Deimling, B. Wullich, and T. Lengauer. Estimating cancer survival and clinical outcome based on genetic tumor progression scores. *Bioinformatics*, 21(10):2438–2446, May 2005.

[11] N. Beerenwinkel, J. Rahnenfuhrer, R. Kaiser, D. Hoffmann, J. Selbig, and T. Lengauer. Mtreemix: a software package for learning and using mixture models of mutagenetic trees. *Bioinformatics*, 21(9):2106–2107, 2005.

[12] N. Beerenwinkel, N. Eriksson, and B. Sturmfels. Conjunctive bayesian networks. *Bernoulli*, 13(4):893–909, Jan 2007.

[13] N. Beerenwinkel, N. Eriksson, and B. Sturmfels. Evolution on distributive lattices. *J Theor Biol*, 242(2):409–20, Sep 2006.

[14] M. Gerstung, M. Baudis, H. Moch, and N. Beerenwinkel. Quantifying cancer progression with conjunctive bayesian networks. *Bioinformatics*, 25(21):2809–15, Nov 2009.

[15] R.E. Tarjan. Finding optimum branchings. *Networks*, 7(1):25–36, 1977.

[16] R.M. Karp. A simple derivation of edmond's algorithm for optimum branching. *Networks*, 1(265-272):5, 1971.

[17] P. Camerini, L. Fratta, and F. Maffioli. The k best spanning arborescences of a network. *Networks*, 10(2):91–110, 1980.

[18] F. Mitelman, B. Johansson, and F. Mertens (Eds.). Mitelman database of chromosome aberrations and gene fusions in cancer, 2010. http://cgap.nci.nih.gov/Chromosomes/Mitelman.

[19] R. Durbin, S.R. Eddy, A. Krogh, and G. Mitchison. *Biological Sequence Analysis*. Cambridge University Press, Cambridge, 1998.

[20] A.P. Dempster, N.M. Laird, and D.B. Rubin. Maximum likelihood from incomplete data via the em algorithm. *J Roy Stat Soc B*, 39(1):1–38, 1977.

[21] M. Höglund, D. Gisselsson, G.B. Hansen, T. Säll, F. Mitelman, and M. Nilbert. Dissecting karyotypic patterns in colorectal tumors: Two distinct but overlapping pathways in the adenoma-carcinoma transition. *Canc Res*, 62:5939–5946, 2002.

[22] M. Höglund, D. Gisselsson, M. Soller, G.B. Hansen, P. Elfving, and F. Mitelman. Dissecting karyotypic patterns in renal cell carcinoma: an analysis of the accumulated cytogenetic data. *Canc Genet Cytogenet*, 153(1):1–9, 2004.

